# A biologically plausible network for the computation of orientation dominance

**Kritika Muralidharan**
Statistical Visual Computing Laboratory
University of California San Diego
La Jolla, CA 92039
krmurali@ucsd.edu

**Nuno Vasconcelos**
Statistical Visual Computing Laboratory
University of California San Diego
La Jolla, CA 92039
nuno@ece.ucsd.edu

## Abstract

The determination of dominant orientation at a given image location is formulated as a decision-theoretic question. This leads to a novel measure for the dominance of a given orientation $\theta$, which is similar to that used by SIFT. It is then shown that the new measure can be computed with a network that implements the sequence of operations of the standard neurophysiological model of V1. The measure can thus be seen as a biologically plausible version of SIFT, and is denoted as bioSIFT. The network units are shown to exhibit trademark properties of V1 neurons, such as cross-orientation suppression, sparseness and independence. The connection between SIFT and biological vision provides a justification for the success of SIFT-like features and reinforces the importance of contrast normalization in computer vision. We illustrate this by replacing the Gabor units of an HMAX network with the new bioSIFT units. This is shown to lead to significant gains for classification tasks, leading to state-of-the-art performance among biologically inspired network models and performance competitive with the best non-biological object recognition systems.

## 1 Introduction

In the past decade, computer vision research in object recognition has firmly established the efficacy of representing images as collections of local descriptors of edge orientation. These descriptors are usually based on histograms of dominant orientation, for example, the edge orientation histograms of [1], the SIFT descriptor of [2], or the HOG features of [3]. SIFT, in particular, could be considered today's default (low-level) representation for object recognition, adopted by hundreds of computer vision papers. The SIFT descriptor is heavily inspired by known computations of the early visual cortex [2], but has no formal detailed connection to computational neuroscience. Interestingly, a parallel, and equally important but seemingly unrelated, development has taken place in this area in the recent past. After many decades of modeling simple cells as linear filters plus "some" nonlinearity [4], neuroscientists have developed a much firmer understanding of their non-linear behavior. One property that has always appeared essential to the robustness of biological vision is the ability of individual cells to adapt their dynamic range to the strength of the visual stimulus. This adaptation appears as early as in the retina [5], is prevalent throughout the visual cortex [6], and seems responsible for the remarkable ability of the visual system to adapt to lighting variations. Within the last decade, it has been explained by the implementation of gain control in individual neurons, through the divisive normalization of their responses by those of their neighbors [7, 8]. Again, hundreds of papers have been written on divisive normalization, and its consequences for visual processing. Today, there appears to be little dispute about its role as a component of the *standard neurophysiological model* of early vision [9].

In this work, we establish a formal connection between these two developments. This connection is inspired by recent work on the link between the computations of the standard model and the basic operations of statistical decision theory [10]. We start by formulating the central motivating question for descriptors such as SIFT or HOG, *how to represent locally dominant image orientation*, as a decision-theoretic problem. An orientation $\theta$ is defined as dominant, at a location $l$ of the visual field, if the Gabor response of orientation $\theta$ at $l$, $x_\theta(l)$, is both *large* and *distinct from those of other orientations*. An *optimal statistical test* is then derived to determine if $x_\theta(l)$ is *distinct* from the responses of remaining orientations. The core of this test is the *posterior probability of orientation of the visual stimulus at $l$, given $x_\theta(l)$*. The dominance of orientation $\theta$, within a neighborhood $\mathcal{R}$, is then defined as the *expected strength of responses $x_\theta(l)$, in $\mathcal{R}$, which are distinct*. This is shown to be a sum of the response amplitudes $|x_\theta(l)|$ across $\mathcal{R}$, with each location weighted by the posterior probability that it contains stimulus of orientation $\theta$.

The resulting representation of orientation is similar to that of SIFT, which assigns each point to a dominant orientation and integrates responses over $\mathcal{R}$. The main difference is that a location could contribute to more than one orientation, since the expected strength relies on a soft assignment of locations to orientations, according to their posterior orientation probability. Exploiting known properties of natural image statistics, and the framework of [10], we then show that this measure of orientation dominance can be *computed with the sequence of operations of the standard neurophysiological model*: simple cells composed of a linear filter, divisive normalization, and a saturating non-linearity, and complex cells that implement spatial pooling. The proposed measure of orientation dominance can then be seen as a biologically plausible version of that used by SIFT, and is denoted by bioSIFT. BioSIFT units are shown to exhibit the trademark properties of V1 neurons: their responses are closely fit by the *Naka-Rushton equation* [11], and they exhibit an inhibitory behavior, known as *cross-orientation suppression*, which is ubiquitous in V1 [12]. We note, however, that our goal is not to provide an alternative to SIFT. On the contrary, the formal connection between findings from computer vision and neuroscience provides additional justification to both the success of SIFT in computer vision, and the importance of divisive normalization in the visual cortex, as well as its connection to the determination of orientation dominance.

The main practical benefit of bioSIFT is to improve the performance of biologically plausible recognition networks, whose performance it brings close to the level of the state of the art in computer vision. In the process of doing this, it points to the importance of divisive normalization in vision. While such normalization tends to be justified as a means to increase robustness to variations of illumination, a hypothesis that we do not dispute, it appears to make a tremendous difference even when such variations do not hold. We illustrate these points through object recognition experiments with HMAX networks [13]. It is shown that the simple replacement of Gabor filter responses with the normalized orientation descriptors of bioSIFT produces very significant gains in recognition accuracy. These gains hold for standard datasets, such as Caltech101, where lighting variations are not a substantial nuisance. This points to the alternative hypothesis that the fundamental role of contrast normalization is to determine orientation dominance. The hypothesis is substantiated by the fact that the bioSIFT enhanced HMAX network substantially outperforms the previous best results in the literature of biologically-inspired recognition networks [14, 15]. While these networks implement a number of operations similar to those of bioSIFT, including the use of contrast normalized units, they do not have a precise functional justification (such as the determination of orientation dominance), lack a well defined optimality criterion, and do not have a rigorous statistical interpretation. The importance of these properties is further illustrated by experiments in a dataset composed exclusively of natural scenes [16], which (unlike Caltech) fully matches the assumptions under which bioSIFT is optimal (natural image statistics). In this dataset, the HMAX network with the bioSIFT features has performance identical to that of very recent state-of-the-art computer vision methods.

## 2  The bioSIFT Features

We start by describing the implementation of the bioSIFT network in detail. We lay out the computations, establish their conformity with the standard neurophysiological model, and analyze the statistical meaning of the computed features.

## 2.1 Motivation

Various authors have argued that perceptual systems compute optimal decisions tuned to the statistics of natural stimuli [17, 18, 19]. The ubiquity of orientation processing in visual cortex suggests that the estimation of local orientation is important for tasks such as object recognition. This is reinforced by the success, in computer vision, of algorithms based on SIFT or SIFT-like descriptors. While the classical view was that the brain simply performs a linear decomposition into orientation channels, through Gabor filtering, SIFT representations emphasize the estimation of *dominant* orientation. The latter is a very non-linear operation, involving the comparison of response strength across orientation channels, and requires inter-channel normalization. In SIFT, this is performed implicitly, by combining the computation of gradients with some post-processing heuristics. More formal estimates of dominant orientation can be obtained by formulating the problem in decision-theoretic terms, and deriving optimal decision rules for its solution. For this, we assume that the visual system infers dominant orientation from a set of visual features $\mathbf{x} \in \mathbb{R}^M$, which measure stimulus amplitude at each orientation. In this work, we assume these features to be the set of responses $X_i = \mathcal{I} \circ \mathcal{G}_i$ of the stimulus $\mathcal{I}$, to a bank of Gabor filters $\mathcal{G}_i$. Here, $\mathcal{G}_i$ is the filter of $i^{th}$ orientation, and $\circ$ convolution. In principle, determining whether there is a dominant orientation requires the joint inspection of all feature channels $X_i$. Statistically, this implies modeling the joint feature distribution and is intractable for low-level vision.

A more tractable question is whether the $i^{th}$ channel responses, $X_i$, are distinct from those of the other channels, $X_j, j \neq i$. Letting $\theta$ denote the channel orientation, i.e. $P_{X|\theta}(x|i) = P_{X_i}(x)$, this question can be posed as a classification problem with two hypotheses of label $Y \in \{0, 1\}$, where

- $Y = 1$ if the $i^{th}$ channel responses are distinct, i.e. $P(X = x, \theta = i) \neq P(X = x, \theta \neq i)$,
- $Y = 0$ otherwise, i.e. $P(X = x, \theta = i) = P(X = x, \theta \neq i)$.

This problem has class-conditional densities

$$P(X = x, \theta = i|Y = 1) = P(X = x, \theta = i) = P_{X|\theta}(x|i)P_\theta(i)$$

$$P(X = x, \theta = i|Y = 0) = P(X = x, \theta \neq i) = \sum_{j \neq i} P_{X|\theta}(x|j)P_\theta(j)$$

and the posterior probability of the 'distinct' hypothesis given an observation from channel $i$ is

$$P(Y = 1|X = x, \theta = i) = \frac{P_{X|\theta}(x|i)P_\theta(i)}{\sum_j P_{X|\theta}(x|j)P_\theta(j)} = P_{\theta|X}(i|x) \tag{1}$$

where we have assumed that $P_Y(0) = P_Y(1) = 1/2$. Given the response $x_i(l)$ of $X_i$ at location $l \in \mathcal{R}$, the minimum probability of error (MPE) decision rule is to declare it distinct when

$$P_{\theta|X}(i|x_i(l)) = \frac{P_{X_i}(x_i(l))P_\theta(i)}{\sum_j P_{X_j}(x_i(l))P_\theta(j)} \geq \frac{1}{2}. \tag{2}$$

While this test determines if the responses of $X_i$ are distinct from those of $X_{j \neq i}$, it does not determine if $X_i$ is dominant: $X_i$ could be distinct because it is the only feature that does not respond to the stimulus in $\mathcal{R}$. The second question is to determine if the responses of $X_i$ are both distinct and large. This requires a new random variable

$$S(x_i) = \begin{cases} |x_i|, & \text{if } Y = 1 \\ 0, & \text{if } Y = 0. \end{cases} \tag{3}$$

which measures the strength (absolute value) of the distinct responses. The expected strength of distinct responses in $\mathcal{R}$ is then

$$E_{Y,X|\theta}[S(X)|\theta = i] = \int |x| P_{Y|X,\theta}(1|x, i) P_{X|\theta}(x|i) dx \tag{4}$$

$$= \int |x| P_{\theta|X}(i|x) P_{X_i}(x) dx. \tag{5}$$

The empirical estimate of (5) from the sample $x_i(l), l \in \mathcal{R}$, is

$$\widehat{S(X_i)}_{\mathcal{R}} = \frac{1}{|\mathcal{R}|} \sum_l |x_i(l)| P_{\theta|X}(i|x_i(l)). \tag{6}$$

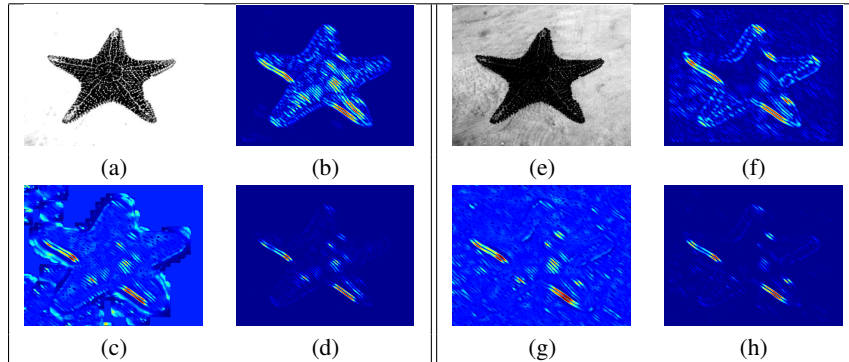

Figure 1: bioSIFT computations for given orientation $\theta$: (a) an image, (b) response of Gabor filter of orientation $\theta$, (c) posterior probability map for orientation $\theta$, (d) orientation dominance measure for channel $\theta$; (e),(f),(g),(h) the image, Gabor response, posterior probability, and dominance measure of same channel for a contrast-reduced version of the image.

This *measure of the dominance of the $i^{th}$ orientation* is a sum of the response amplitudes $|x_i(l)|$ across $\mathcal{R}$, with each location weighted by the posterior probability that it contains stimulus of that orientation. It is similar to the measure used by SIFT, which assigns each point to a dominant orientation and integrates responses over $\mathcal{R}$. The main difference is that a location could contribute to more than one orientation, since the expected strength relies on a soft assignment of locations to orientations, according to their posterior orientation probability.

Figure 1 illustrates the computations of (6) for the image shown in a). The response of a Gabor filter of orientation $\theta = 3\pi/4$ is shown in b), and the orientation probability map $P_{\theta|X}(i|x_i)$ in c). Note that these probabilities are much smaller than the Gabor responses in the body of the starfish, where the image is textured but there is no significant structure of orientation $\theta$. On the other hand, they are largest for the locations where the orientation is dominant. Figure 1 d) shows the final dominance measure. The combined multiplication by the Gabor responses and averaging over $\mathcal{R}$ magnifies the responses where the orientation is dominant, suppressing the details due to texture or noise. This can be seen by comparing b) and d). Overall, (6) is large when the $i^{th}$ orientation responses are 1) distinct from those of other channels and 2) large. It is small when they are either indistinct or small. One interesting property is that it penalizes large responses of $X_i$ that are not informative of the presence of stimuli with orientation $i$. Hence, increasing the stimulus contrast does not increase $\widehat{S(X_i)}_{\mathcal{R}}$ when responses $x_i(l)$ cannot be confidently assigned to the $i^{th}$ orientation. This can be seen in Figure 1 f) and h), where the Gabor response and dominance measure are shown for a low-contrast replica of the image of a). While the Gabor responses at low (f) and high (b) contrasts are substantially different, the dominance measure (d and h) stays almost constant. It follows that (6) implements contrast normalization, a topic to which we will return in later sections. It is worth noting that such normalization is accomplished without modeling joint distributions of response across orientations. On the contrary, all quantities in (6) are scalar.

## 3 Biological plausibility

In this section we study the biological plausibility of the orientation dominance measure of (6).

### 3.1 Natural image statistics

Extensive research on the statistics of natural images has shown that the responses of bandpass features follow the generalized gaussian distribution (GGD)

$$P_X(x; \alpha, \beta) = \frac{\beta}{2\alpha\Gamma(1/\beta)} \exp\left(-\left(\frac{|x|^\beta}{\alpha}\right)\right) \qquad (7)$$

where $\Gamma(z) = \int_0^\infty e^{-t}t^{z-1}\,dt, t > 0$ is the Gamma function, $\alpha$ is a scale and $\beta$ a shape parameter. The biological plausibility of statistical inference for GGD stimuli was extensively studied in

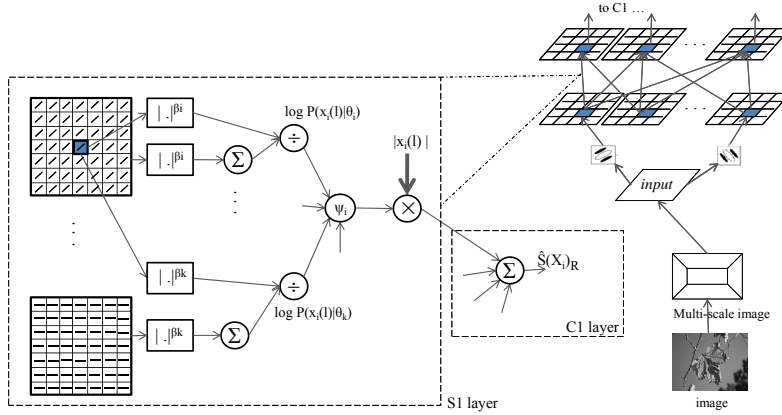

Figure 2: One channel of the bioSIFT network. The large dashed box implements the computations of the simple cell, and the small one those of the complex cell. The simple cell computes the contribution of channel $i$ to the expected value of the dominant response at pixel $x$, indicated by a filled box. Spatial pooling by the complex cell determines the channel's contribution to the expected value of the dominant response within the pooling neighborhood.

[10]. This work shows that various fundamental computations in statistics can indeed be computed biologically when a maximum a posteriori (MAP) estimate is adopted for $\alpha^\beta$, using a conjugate (Gamma) prior. This MAP estimate is

$$\alpha_{MAP} = \left[\frac{\beta}{n+\eta}\left(\sum_{j=1}^{n}|x(j)|^\beta + \nu\right)\right]^{1/\beta} \tag{8}$$

where $\nu$ and $\eta$ are the prior hyperparameters, and $x(j)$ a sample of training points. As is usual in Bayesian inference, the hyperparameter values are important when the sample is too small to enable reliable inference. This is not the case for the current work, where the estimates remain constant over a substantial range of their values. Hence, we simply set $\nu = 10^{-3}$ and $\eta = 1$ in all experiments. For natural images, the value of $\beta$ is quite stable. We use $\beta = 0.5$, (determined by fitting the GGD to a large set of images) in our experiments.

## 3.2 Biological computations

To derive a biologically plausible form of (6) we start by assuming that $P_\theta(i) = \frac{1}{M}$. This is mostly for simplicity, the discussion could be generalized to account for any prior distribution of orientations. Under this assumption, using (1)

$$P_{\theta|X}(\theta = j|x) = \frac{P_{X|\theta}(x|\theta = j)}{\sum_k P_{X|\theta}(x|\theta = k)} = \frac{P_{X_j}(x)}{\sum_k P_{X_k}(x)} \tag{9}$$

and

$$\widehat{S(X_i)}_\mathcal{R} \quad \propto \quad \sum_{l \in \mathcal{R}} |x_i(l)| \psi_i \left[\log P_{X_1}(x_i(l)), \ldots, \log P_{X_M}(x_i(l))\right] \tag{10}$$

where $\psi_k$ is the classical softmax activation function

$$\psi_k(q_1, ..., q_n) = \frac{\exp(q_k)}{\sum_{j=1}^{n} \exp(q_j)}, \tag{11}$$

$q_j$ the log-likelihood (up to constants that cancel in (11))

$$q_j = \log P_{X_j}(x_i(l)) = -\phi(x_i(l); \theta_j) - K_j \tag{12}$$

and, from (7) with the MAP estimate of $\alpha^\beta$ from (8) and the responses in $\mathcal{R}$ as training sample,

$$\phi(x; \theta_k) = \frac{|x|^\beta}{\xi_k}; \xi_k = \frac{\beta}{|\mathcal{R}| + \eta}\left(\sum_{l \in \mathcal{R}} |x_k(l)|^\beta + \nu\right); K_j = \log \alpha_j = \frac{1}{\beta} \log \xi_j. \tag{13}$$

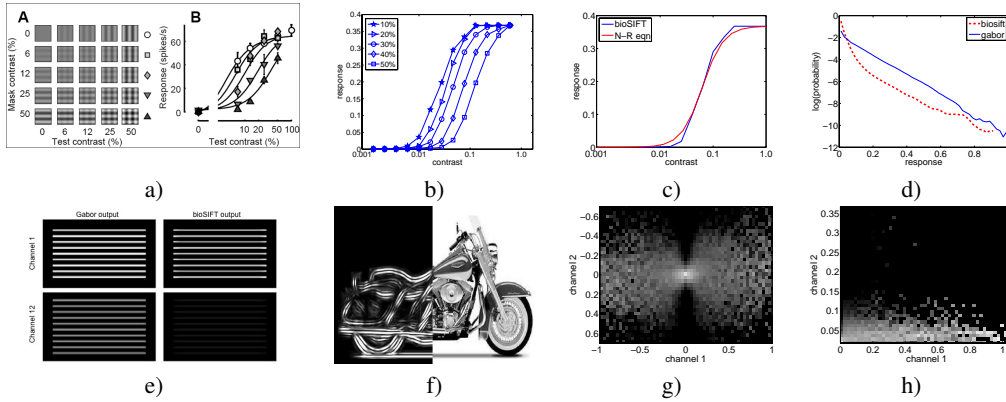

Figure 3: (a) COS in real neurons(from [12]), and (b) in bioSIFT features (c) Contrast response in bioSIFT features and corresponding Naka-Rushton fit (d)distributions of Gabor and bioSIFT amplitudes (e) Example of Orientation selectivity (f) sample image and maximum biosift response at each location (g,h) conditional histograms of adjacent channels for Gabor(g) and bioSIFT(h) features.

The computations of (11)-(13) are those performed by simple cells in the standard neurophysiological model of V1. A bank of linear filters is applied at each location $l$ of the field of view. This produces the Gabor responses $x_i(l)$. Each response $x_i(l)$ is divisively normalized by the sum of responses in the neighborhood $\mathcal{R}$, for each orientation channel $k$, using (13). Notice that this implies that the conditional distribution of responses of a channel is learned locally, from the sample of responses in $\mathcal{R}$. Altogether, (12) implements the computations of a divisively normalized simple cell. Finally, the softmax $\psi_k$ is a multi-way sigmoidal non-linearity which replicates the well known saturating behavior of simple cells. The computation of the orientation dominance measure by (10) then corresponds to a complex cell, which pools the simple cell responses in $\mathcal{R}$, modulated by the magnitude of the underlying Gabor responses. This produces each channel's contribution to the bioSIFT descriptor. A graphical description of the network is presented in Figure 2.

### 3.3   Naka-Rushton fit

In addition to replicating the standard model of V1, the biological plausibility of the bioSIFT features can be substantiated by checking if they reproduce well-established properties of neuronal responses. One characteristic property of neural responses of monkey and cat V1 is the tightness with which they can be fit by the Naka-Rushton equation [11]. The equation describes the average response to a sinusoidal grating of contrast $c$ as

$$R = R_{max}\frac{c^q}{c_{50}^q + c^q} \tag{14}$$

where $R_{max}$ is the maximum mean response, $c_{50}$ is the *semi-saturation contrast* i.e. the contrast at which the response is half the saturation value. The parameter $q$, which determines the steepness of the curve, is remarkably stable for V1 neurons, where it takes values around 2 [20]. The fit between the contrast response of a bioSIFT unit and the Naka-Rushton function was determined, using the procedure of [11], and is shown in Figure 3 c). As in biology, the Naka-Rushton model fits the bioSIFT data quite well. Over multiple trials, the $q$ parameter for the best fitting curve is stable and stays in the interval $(1.7, 2.1)$.

### 3.4   Inhibitory effects

It is well known that V1 neurons have a characteristic inhibitory behavior, known as *cross-orientation suppression* (COS) [12, 7, 21]. This suppression is observed by measuring the response of a neuron, tuned to an orientation $\theta$, to a sinusoidal grating of orthogonal orientation ($\theta \pm 90°$). When presented by itself, the grating barely evokes a response from the neuron. However, if superimposed with a grating of another orientation, it significantly reduces the response of the neuron to the latter. To test if the bioSIFT features exhibit COS, we repeated the set of experiments reported

in [12]. These consist of measuring a simple cell response to a set of sinusoidal plaids obtained by summing 1) a test grating oriented along the cell's preferred orientation, and 2) a mask grating of orthogonal orientation. The test and the mask have the same frequency as the cell's Gabor filter. The cell response is recorded as a function of the contrast of the gratings. Figure 3 a) shows the results reported in [12], for a real neuron. The stimuli are shown on the left and the neuron's response on the right. Note the suppression of the latter when the mask contrast increases. The response of the bioSIFT simple cell, shown in Figure 3 b), is identical to that of the neuron.

From a functional point of view, the great advantage of COS is the resulting increase in selectivity of the orientation channels. This is illustrated in Figure 3 (e). The figure shows the results of an experiment that measured the response of 12 Gabor filters of orientation in $[0^o, 180^o]$ to a horizontal grating. While both the first and twelfth Gabor filters have relatively large responses to this stimulus, the twelfth channel of bioSIFT is strongly suppressed. When combined with the contrast invariance of Figure 1, this leads to a representation with strong orientation discrimination and robustness to lighting variations. An example of this is shown in Figure 3 (f) which shows the value of the dominance measure for the most dominant orientation at each image location (in "split screen" with the original image). Note how the bioSIFT features capture information about dominant orientation and object shape, suppressing uninformative or noisy pixels.

### 3.5    Independence and sparseness

Barlow [18] argued that the goal of sensory systems is to reduce redundancy, so as to produce statistically independent responses. A known property of the responses of bandpass features to natural images is a consistent pattern of higher order dependence, characterized by bow-tie shaped conditional distributions between feature pairs. This pattern is depicted in Figure 3 g), which shows the histogram of responses of a Gabor feature, conditioned on the response of the co-located feature of an adjacent orientation channel. Simoncelli [22] showed that divisively normalizing linear filter responses reduces these higher-order dependencies, making the features independent. As can be seen from (10), (12), and (13), the bioSIFT network divisively normalizes each Gabor response by the sum, across the spatial neighborhood $\mathcal{R}$, of responses from each of the Gabor orientations (11). It is thus not surprising that, as shown in Figure 3 h), the conditional histograms of bioSIFT features are zero outside a small horizontal band around the horizontal axis. This implies that they are independent (knowledge of the value of one feature does not modify the distribution of responses of the other).This is a consistent observation across bioSIFT feature pairs.

Another important, and extensively researched, property of V1 responses is their sparseness. Channel sparseness is closely related to independence across channels. Sparse representations have several important advantages, such as increased generalization ability and energy efficiency of neural decision-making circuits. Given the discussion above, it is not surprising that the contrast normalization inherent to the bioSIFT representation also makes it more sparse. This is shown in Figure 3 d), which compares the sparseness of the responses of both a Gabor filter and a bioSIFT unit to a natural image. It is worth noting that these properties have not been exploited in the SIFT literature itself. For example, independence could lead to more efficient implementations of SIFT-based recognizers than the standard visual words approach, which requires an expensive quantization of SIFT features with respect to a large codebook. We leave this as a topic for future research.

## 4    Experimental Evaluation

In this section, we report on experiments designed to evaluate the benefits, for recognition, of the connections between SIFT and the standard neurophysiological model.

### 4.1    Biologically inspired object recognition

Biologically motivated networks for object recognition have been recently the subject of substantial research [13, 23, 14, 15]. To evaluate the impact of adding bioSIFT features to these networks, we considered the HMAX network of [13], which mimics the structure of the visual cortex as a cascade of alternating simple and complex cell layers. The first layer encodes the input image as a set of complex cell responses, and the second layer measures the distance between these responses and a set of learned prototypes. The vector of these distances is then classified with a linear SVM.

| Model | 30 training images/cat. |
|---|---|
| Base HMAX [13] | 42 |
| + enhancements [23] | 56 |
| Pinto et al. [14] | 65 |
| Jarrett et al [15] | 65.5 |
| Lazebnik et al. [16] | $64.6 \pm 0.8$ |
| Zhang et al. [24] | $66.2 \pm 0.5$ |
| NBNN [25] | 70.4 |
| Yang et al. [26] | $73.2 \pm 0.5$ |
| **base bioSIFT HMAX** | 54.5 |
| **+enhancements** | $69.3 \pm 0.3$ |

| Model | Performance |
|---|---|
| Fei-Fei et al [27] | 65.2 |
| Lazebnik et al. [16] | $81.4 \pm 0.5$ |
| Yang et al [26] | $80.3 \pm 0.9$ |
| Kernel Codebooks [28] | $76.7 \pm 0.4$ |
| HMAX with bioSIFT | $80.1 \pm 0.6$ |

Figure 4: Classification Results on Caltech-101(left) and the Scene Classification Database(right)

For this evaluation, each unit of the first layer was replaced by a bioSIFT unit, implemented as in Figure 2. The experimental setup is similar to that of [23]: multi-class classification on Caltech101 (with the size of the images reduced so that their height is 140) using 30 images/object for training and at-most 50 for testing. The baseline accuracy of [13] was $42\%$. The work of [23] introduced several enhancements that were shown to considerably improve this baseline. Two of these enhancements, sparsification and inhibition, were along the lines of the contributions discussed in this work. Others, such as limiting receptive fields to restrict invariance, and discriminant selection of prototypes could also be combined with bioSIFT. The base performance of the network with bioSIFT ($54.5\%$) is superior to that of all comparable extensions of [23] ($49\%$). This can be attributed to the fact that those extensions are mostly heuristic, while those now proposed have a more sound decision-theoretic basis. In fact, the simple addition of bioSIFT features to the HMAX network outperforms all extensions of [23] up to the prototype selection stage ($54\%$). When bioSIFT is complemented with limited C2 invariance and prototype selection the performance improves to $69\%$, which is better than all results from [23]. In fact, the HMAX network with bioSIFT outperforms the state-of-the-art [1] performance ($65.5\%$) for biologically inspired networks [15]. This improvement is interesting, given that these networks also implement most of the operations of the bioSIFT unit (filtering, normalization, pooling, saturation, etc.). The main difference is that this is done without a clear functional justification, optimality criteria, or statistical interpretation. In result, the sequence of operations is not the same, there is no guarantee that normalization provides optimal estimates of orientation dominance, or even that it corresponds to optimal statistical learning, as in (8).

## 4.2 Natural scene classification

When compared to the state-of-the-art from the computer vision literature, the HMAX+bioSIFT network, does not fare as well. Most notably, it has worse performance than the method of Yang et al. [26], which holds the current best results for this dataset (single descriptor methods). This is explained by two main reasons. The first is that the networks are not equivalent. Yang et. al rely on a sparse coding representation in layer 2, which is likely to be more effective than the simple Gaussian units of HMAX. This problem could be eliminated by combining bioSIFT with the same sparse representation, something that we have not attempted. A second reason is that bioSIFT is not exactly optimal for Caltech, because this dataset contains various classes with many non-natural images. To avoid this problem, we have also evaluated the bioSIFT features on the scene classification task of [16]. Using the same HMAX setup, a simple linear classifier and 3000 layer 2 units, the network achieves a classification performance of $80.1\%$ (see Figure 4). This is a substantial improvement, since these results are nearly identical to those of Yang et al. [26], and better than many of those of other methods from the computer vision literature. Overall, these results suggest that orientation dominance is an important property for visual recognition. In particular, the improved performance of the bioSIFT units cannot be explained by the importance of contrast normalization, since this is not a major nuisance for the datasets considered, it is also implemented by the other networks, bioSIFT is not optimized to normalize contrast, and it is unlikely that constrast variations would be more of an issue on Caltech than on the natural scene dataset.

## Footnotes

[1][14] reports $65\%$, but for a network with a much larger number of units (SVM dimension) than what is used by all other networks. Our implementation of their network with comparable parameters only achieved $42\%$.

# References

[1] W. T. Freeman and M. Roth, "Orientation histograms for hand gesture recognition," in *IEEE Intl. Wkshp. on Automatic Face and Gesture Recognition*, 1995.

[2] D. G. Lowe, "Distinctive image features from scale-invariant keypoints," *IJCV*, vol. 60(2), pp. 91–110, 2004.

[3] N. Dalal and B. Triggs, "Histograms of oriented gradients for human detection," in *Proc. IEEE Conf. CVPR*, 2005.

[4] D. H. Hubel and T. N. Wiesel, "Receptive fields, binocular interaction, and functional architecture in the cat's visual cortex," *Journal of Physiology*, vol. 160, 1962.

[5] R. Shapley and J. D. Victor, "The contrast gain control of the cat retina," *Vision Research*, vol. 19, pp. 431–434, 1979.

[6] S. E. Palmer, *Vision Science: Photons to Phenomenology*. The MIT Press, 1999.

[7] D. Heeger, "Normalization of cell responses in cat striate cortex," *Visual Neuroscience*, vol. 9, 1992.

[8] M. Carandini, D. J. Heeger, and J. A. Movshon, "Linearity and normalization in simple cells of macaque primary visual cortex," *Journal of Neuroscience*, vol. 17, pp. 8621–8644, 1997.

[9] M. Carandini, J. B. Demb, V. Mante, D. J. Tolhurst, Y. Dan, B. A. Olshausen, J. L. Gallant, and N. C. Rust, "Do we know what the early visual system does?," *Journal of Neuroscience*, vol. 25, 2005.

[10] D. Gao and N. Vasconcelos, "Decision-theoritic saliency: computational principles, biological plausibility, and implications for neurophysiology and psychophysics," *Neural Computation*, vol. 21, 2009.

[11] M. Chirimuuta and D. J. Tolhurst, "Does a bayesian model of v1 contrast coding offer a neurophysiological account of contrast discrimination?," *Vision Research*, vol. 45, pp. 2943–2959, 2005.

[12] M. Carandini, *Receptive fields and suppressive fields in the early visual system*. MIT Press, 2004.

[13] T. Serre, L. Wolf, and T. Poggio, "Object recognition with features inspired by visual cortex," in *IEEE Conf. CVPR*, 2005.

[14] N. Pinto, D. Cox, and J. Dicarlo, "Why is real-world visual object recognition hard?," *PLoS Computational Biology*, 2008.

[15] K. Jarrett, K. Kavukcuoglu, M. Ranzato, and Y. Lecun, "What is the best multi-stage architecture for object recognition?," in *Proc. IEEE International Conference on Computer Vision*, 2009.

[16] S. Lazebnik, C. Schmid, and J. Ponce, "Beyond bags of features: Spatial pyramid matching for recognizing natural scene categories," in *CVPR*, 2006.

[17] F. Attneave, "Informational aspects of visual perception," *Psychological review*, vol. 61, pp. 183–193, 1954.

[18] H. B. Barlow, "Redundancy reduction revisited," *Network: Computation in Neural Systems*, vol. 12, 2001.

[19] D. C. Knill and W. Richards, *Perception as Bayesian inference*. Cambridge University Press, 1996.

[20] D. G. Albrecht and D. B. Hamilton, "Striate cortex of monkey and cat: contrast response function," *Journal of Neurophysiology*, vol. 48, pp. 217–237, 1982.

[21] M. C. Morrone, D. C. Burr, and L. Maffei, "Functional implications of cross orientation inhibition of cortical visual cells i. neurophysiological evidence," *Proc. Royal Society London B*, vol. 216, pp. 335–354, 1982.

[22] M. J. Wainwright, O. Schwartz, and E. P. Simoncelli, "Natural image statistics and divisive normalization: Modeling nonlinearities and adaptation in cortical neurons," in *Probabilistic Models of the Brain: Perception and Neural Function*, pp. 203–222, MIT Press, 2002.

[23] J. Mutch and D. Lowe, "Object class recognition and localization using sparse features with limited receptive fields," *IJCV*, vol. 80, pp. 45–57, 2008.

[24] H. Zhang, A. Berg, M. Maire, and J. Malik, "Svm-knn: Discriminative nearest neigbor classification for visual category recognition," in *Proc. IEEE Conf CVPR*, 2006.

[25] O. Boiman, E. Shechtman, and M. Irani, "In defense of nearest-neighbor based image classification," in *Proc. IEEE Conf. CVPR*, 2008.

[26] J. Yang, K. Yu, Y. Gong, and T. Huang, "Linear spatial pyramid matching using sparse coding for image classification," in *Proc. IEEE Conf. CVPR*, 2009.

[27] L. Fei-Fei and P. Perona, "A bayesian heirarchical model for learning natural scene categories," in *Proc. IEEE Conf CVPR*, 2005.

[28] J. C. van Gemert, J. M. Geusebroek, C. J. Veenman, and A. W. M. Smeulders, "Kernel codebooks for scene categorisation," in *Proc ECCV*, 2008.

